# Refractoriness and Neural Precision

**Michael J. Berry II  and  Markus Meister**
Molecular and Cellular Biology Department
Harvard University
Cambridge, MA 02138

## Abstract

The relationship between a neuron's refractory period and the precision of its response to identical stimuli was investigated. We constructed a model of a spiking neuron that combines probabilistic firing with a refractory period. For realistic refractoriness, the model closely reproduced both the average firing rate and the response precision of a retinal ganglion cell. The model is based on a "free" firing rate, which exists in the absence of refractoriness. This function may be a better description of a spiking neuron's response than the peri-stimulus time histogram.

## 1  INTRODUCTION

The response of neurons to repeated stimuli is intrinsically noisy. In order to take this trial-to-trial variability into account, the response of a spiking neuron is often described by an instantaneous probability for generating an action potential. The response variability of such a model is determined by Poisson counting statistics; in particular, the variance in the spike count is equal to the mean spike count for any time bin (Rieke, 1997). However, recent experiments have found far greater precision in the vertebrate retina (Berry, 1997) and the H1 interneuron in the fly visual system (de Ruyter, 1997). In both cases, the neurons exhibited sharp transitions between silence and nearly maximal firing. When a neuron is firing near its maximum rate, refractoriness causes spikes to become more regularly spaced than for a Poisson process with the same firing rate. Thus, we asked the question: does the refractory period play an important role in a neuron's response precision under these stimulus conditions?

## 2  FIRING EVENTS IN RETINAL GANGLION CELLS

We addressed the role of refractoriness in the precision of light responses for retinal ganglion cells.

### 2.1  RECORDING AND STIMULATION

Experiments were performed on the larval tiger salamander. The retina was isolated from the eye and superfused with oxygenated Ringer's solution. Action potentials from retinal

ganglion cells were recorded extracellularly with a multi-electrode array, and their spike times measured relative to the beginning of each stimulus repeat (Meister, 1994). Spatially uniform white light was projected from a computer monitor onto the photoreceptor layer. The intensity was flickered by choosing a new value at random from a Gaussian distribution (mean $I$, standard deviation $\delta I$) every 30 ms. The mean light level ($I = 4 \cdot 10^{-3}$ W/m$^2$) corresponded to photopic (daylight) vision. Contrast $C$ is defined here as the temporal standard deviation of the light intensity divided by the mean, $C = \delta I / I$. Recordings extended over 60 repeats of a 60-sec segment of random flicker.

The qualitative features of ganglion cell responses to random flicker stimulation at 35 % contrast are seen in Fig. 1. First, spike trains had extensive periods in which no spikes were seen in 60 repeated trials. Many spike trains were sparse, in that the silent periods covered a large fraction of the total stimulus time. Second, during periods of firing, the peri-stimulus time histogram (PSTH) rose from zero to the maximum firing rate (~200 Hz) on a time scale comparable to the time interval between spikes (~10 ms). We have argued that these responses are better viewed as a set of discrete firing "events" than as a continuously varying firing rate (Berry, 1997). In general, the firing events were bursts containing more than one spike (Fig. 1B). Identifiable firing events were seen across cell types; similar results were also found in the rabbit retina (Berry, 1997).

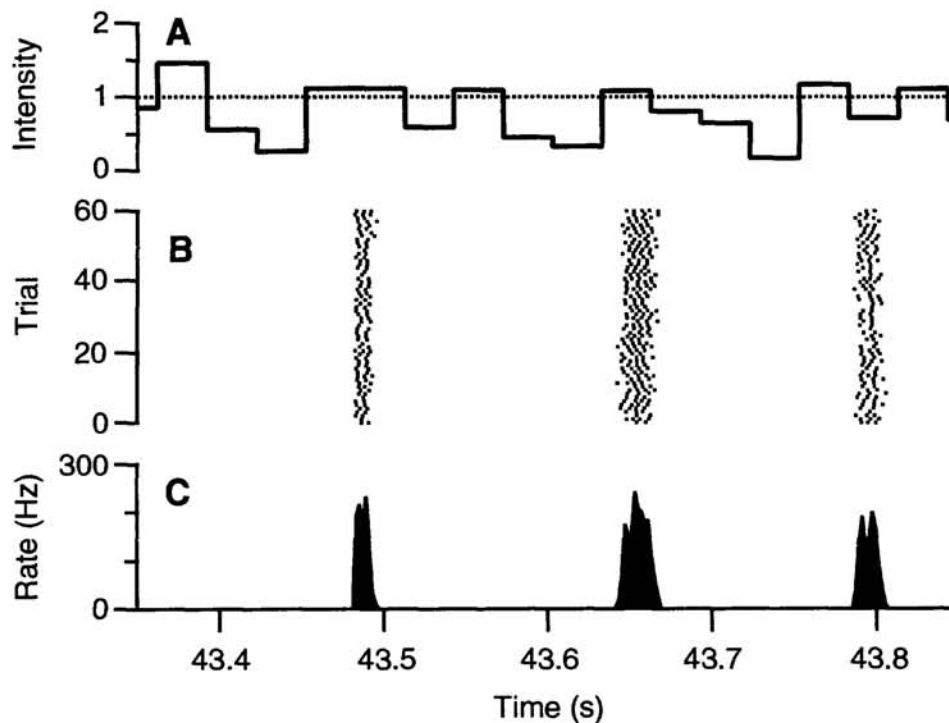

Figure 1: Response of a salamander ganglion cell to random flicker stimulation. (A) Stimulus intensity in units of the mean for a 0.5-s segment, (B) spike rasters from 60 trials, and (C) the firing rate $r(t)$.

## 2.2 FIRING EVENT PRECISION

Discrete episodes of ganglion cell firing were recognized from the PSTH as a contiguous period of firing bounded by periods of complete silence. To provide a consistent demarcation of firing events, we drew the boundaries of a firing event at minima $v$ in the PSTH that were significantly lower than neighboring maxima $p_1$ and $p_2$, such that $\sqrt{p_1 p_2}/v \geq \phi$ with 95 % confidence (Berry, 1997). With these boundaries defined, every spike in each trial was assigned to exactly one firing event.

Measurements of both timing and number precision can be obtained if the spike train is parsed into such firing events. For each firing event $i$, we accumulated the distribution of spike times across trials and calculated several statistics: the average time $T_i$ of the first spike in the event and its standard deviation $\delta T_i$ across trials, which quantified the temporal jitter of the first spike; similarly, the average number $N_i$ of spikes in the event and its variance $\delta N_i^2$ across trials, which quantified the precision of spike number. In trials that contained zero spikes for event $i$, no contribution was made to $T_i$ or $\delta T_i$, while a value of zero was included in the calculation of $N_i$ and $\delta N_i^2$.

For the ganglion cell shown in Fig. 1, the temporal jitter $\delta T$ of the first spike in an event was very small (1 to 10 ms). Thus, repeated trials of the same stimulus typically elicit action potentials with a timing uncertainty of a few milliseconds. The temporal jitter of all firing events was distilled into a single number $\tau$ by taking the median o··er all events. The variance $\delta N^2$ in the spike count was remarkably low as well: it often approached the lower bound imposed by the fact that individual trials necessarily produce integer spike counts. Because $\delta N^2 << N$ for all events, ganglion cell spike trains cannot be completely characterized by their firing rate (Berry, 1997). The spike number precision of a cell was assessed by computing the average variance over events and dividing by the average spike count: $F = \langle \delta N^2 \rangle / \langle N \rangle$. This quantity, also known as the Fano factor, has a value of one for a Poisson process with no refractoriness.

## 3 PROBABILISTIC MODELS OF A SPIKE TRAIN

We start by reviewing one of the simplest probabilistic models of a spike train, the inhomogeneous Poisson model. Here, the measured spike times $\{t_i\}$ are used to estimate the instantaneous rate $r(t)$ of spike generation during a time $\Delta t$. This can be written formally as

$$ r(t) = \frac{1}{M \, \Delta t} \, \Sigma_i \, \Theta(t_i - t) \, \Theta(t + \Delta t - t_i) $$

where $M$ is the number of repeated stimulus trials and $\Theta(x)$ is the Heaviside function

$$ \Theta(x) = \left. \begin{matrix} 1 & x \geq 0 \\ 0 & x < 0 \end{matrix} \right\} \quad . $$

We can randomly generate a sequence of spike trains from a set of random numbers between zero and one: $\{\alpha_i\}$ with $\alpha_i \in (0,1]$. If there is a spike at time $t_i$, then the next spike time $t_{i+1}$ is found by numerically solving the equation

$$ -\ln \alpha_{i+1} = \int_{t_i}^{t_{i+1}} r(t) \, dt \quad . $$

### 3.1 INCLUDING AN ABSOLUTE REFRACTORY PERIOD

In order to add refractoriness to the Poisson spike-generator, we expressed the firing rate as the product of a "free" firing rate $q(t)$, which obtains when the neuron is not refractory, and a recovery function $w(t)$, which describes how the neuron recovers from refractoriness (Johnson, 1983; Miller, 1985). When the recovery function is zero, spiking is not possible; and when it is one, spiking is not affected. The modified rule for selecting spikes then becomes

$$ -\ln \alpha_{i+1} = \int_{t_i}^{t_{i+1}} q(t) w(t - t_i) \, dt \quad . $$

For an absolute refractory period of time $\mu$, the weight function is zero for times between 0 and $\mu$ and one otherwise

$$w(t;\mu) = 1 - \Theta(t)\Theta(\mu - t) \quad .$$

Because the refractory period may exclude spiking in a given time bin, the probability of firing a spike when not prevented by the refractory period is higher than predicted by $r(t)$. This free firing rate $q(t;\mu)$ can be estimated by excluding trials where the neuron is unable to fire due to refractoriness

$$q(t;\mu) = \frac{r(t)}{1 - \frac{1}{M}\Sigma_i\left[1 - w(t - t_i;\mu)\right]} \quad .$$

The sum is restricted to spike times $t_i$ nearest to the time bin on a given trial. This restriction follows from the assumption that the recovery function only depends on the time since the last action potential. Notice that this new probability obeys the inequality $q(t) \geq p(t)$ and also that it depends upon the refractory period $\mu$.

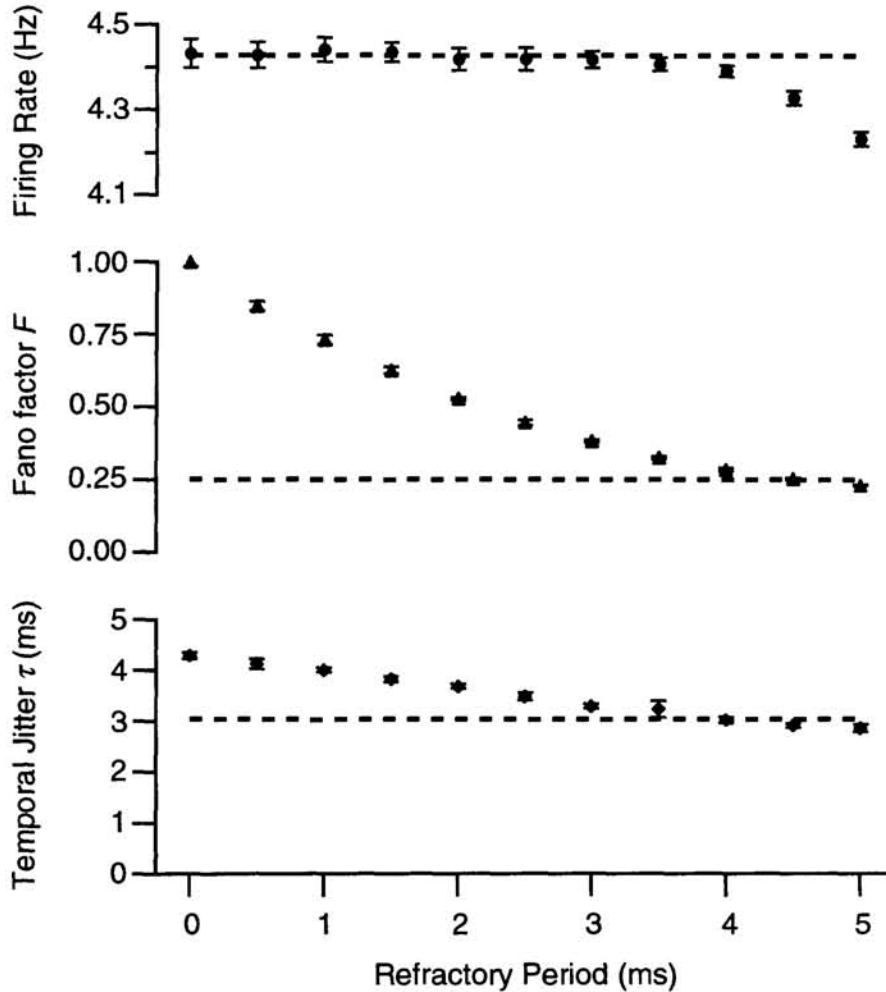

Figure 2: Results for model spike trains with an absolute refractory period. (A) Mean firing rate averaged over a 60-s segment (circles), (B) Fano factor $F$, a measure of spike number precision in an event (triangles), and (C) temporal jitter $\tau$ (diamonds) plotted versus the absolute refractory period $\mu$. Shown in dotted in each panel is the value for the real data.

With this definition of the free firing rate, we can now generate spike trains with the same first order statistics (i.e., the average firing rate) for a range of values of the refractory period $\mu$. For each value of $\mu$, we can then compare the second order statistics (i.e., the precision) of the model spike trains to the real data. To this end, the free rate $q(t)$ was

calculated for a 60-s segment of the response to random flicker of the salamander ganglion cell shown in Fig. 1. Then, $q(t)$ was used to generate 60 spike trains. Firing events were identified in the set of model spike trains, and their precision was calculated. Finally, this procedure was repeated 10 times for each value of the refractory period.

Figure 2A plots the firing rate (circles) generated by the model, averaged over the entire 60-s segment of random flicker with error bars equal to the standard deviation of the rate among the 10 repeated sets. The firing rate of the model matches the actual firing rate for the real ganglion cell (dashed) up to refractory periods of $\mu \approx 4$ ms, although the deviation for larger refractory periods is still quite small. For large enough values of the absolute refractory period, there will be inter-spike intervals in the real data that are shorter than $\mu$. In this case, the free firing rate $q(t)$ cannot be enhanced enough to match the observed firing rate.

While the mean firing rate is approximately constant for refractory periods up to 5 ms, the precision changes dramatically. Figure 2B shows that the Fano factor $F$ (triangles) has the expected value of 1 for no refractory period, but drops to ~ 0.2 for the largest refractory period. In Fig. 2C, the temporal jitter $\tau$ (diamonds) also decreases as refractoriness is added, although the effect is not as large as for the precision of spike number. The sharpening of temporal precision is due to the fact that the probability $q(t)$ rises more steeply than $r(t)$ (see Fig. 4), so that the first spike occurs over a narrower range of times. The number precision of the model matches the real data for $\mu = 4$ to 4.5 ms and the timing precision matches for $\mu \approx 4$ ms. Therefore, a probabilistic spike generator with an absolute refractory period can match both the average firing rate and the precision of a retinal ganglion cell's spike train with roughly the same value of one free parameter.

## 3.2  USING A RELATIVE REFRACTORY PERIOD

Salamander ganglion cells typically have a relative refractory period that lasts beyond their absolute refractory period. This can be seen in Fig. 3A from the distribution of inter-spike intervals $P(\Delta)$ for the ganglion cell shown above – the absolute refractory period lasts for only 2 ms, while relative refractoriness extends to ~ 5 ms. We can include the effects of relative refractoriness by using weight values in $w(t)$ that are between zero and one. Figure 3 illustrates a parameter-free method for determining this weight function. If there were no refractoriness and a neuron had a constant firing rate $q$, then the inter-spike interval distribution would drop exponentially. This behavior is seen from the curve fit in Fig. 3A for intervals in the range 5 to 10 ms. The recovery function $w(t)$ can then be found from the inter-spike interval distribution (Berry, 1998)

$$w(t) = \frac{1}{q} \frac{P(t)}{1 - \int_o^t P(\Delta)d\Delta} \quad .$$

Notice in Fig. 3B, that the recovery function $w(t)$ is zero out to 3 ms, rises almost linearly between 3 and 5 ms, and then reaches unity beyond 5 ms.

Using the weight function shown in Fig. 3B, the free firing rate $q(t)$ was calculated and 10 sets of 60 spike trains were generated. The results, summarized in Table 1, give very close agreement with the real data:

Table 1: Results for a Relative Refractory Period

| QUANTITY | REAL DATA | MODEL | STD. DEV. |
|---|---|---|---|
| Firing Rate | 4.43 Hz | 4.44 Hz | 0.017 Hz |
| Timing Precision $\tau$ | 3.20 ms | 2.95 ms | 0.09 ms |
| Number Precision $F$ | 0.250 | 0.266 | 0.004 |

Thus, a Poisson spike generator with a relative refractory period reproduces the measured precision. A similar test, performed over a population of ganglion cells, also yielded close agreement (Berry, 1998).

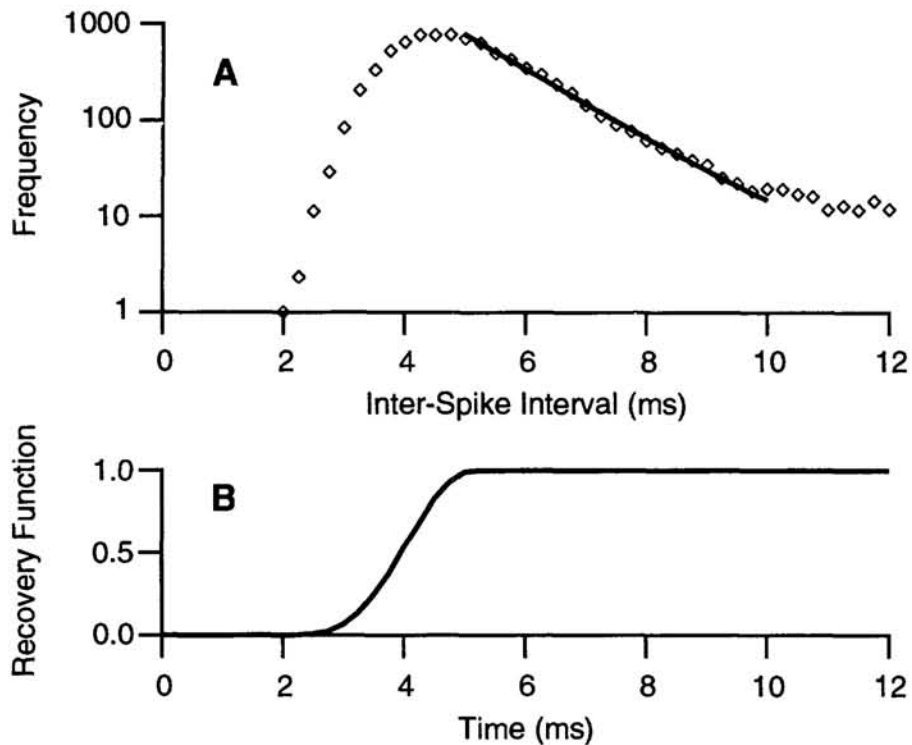

Figure 3: Determination of the relative refractory period. (A) The inter-spike interval distribution (diamonds) is fit by an exponential curve (solid), resulting in (B) the recovery function.

Not only is the average firing rate well-matched by the model, but the firing rate in each time bin is also very similar. Figure 4A compares the firing rate for the real neuron to that generated by the model. The mean-squared error between the two is 4%, while the counting noise, estimated as the variance of the standard error divided by the variance of $r(t)$, is also 4%. Thus, the agreement is limited by the finite number of repeated trials. Figure 4B compares the free firing rate $q(t)$ to the observed rate firing $r(t)$. $q(t)$ is equal to $r(t)$ at the beginning of a firing event, but becomes much larger after several spikes have occurred. In addition, $q(t)$ is generally smoother than $r(t)$, because there is a greater enhancement in $q(t)$ at times following a peak in $r(t)$.

In summary, the free firing rate $q(t)$ can be calculated from the raw spike train with no more computational difficulty than $r(t)$, and thus can be used for any spiking neuron. Furthermore, $q(t)$ has some advantages over $r(t)$: 1) in conjunction with a refractory spike-generator, it produces the correct response precision; 2) it does not saturate at high firing rates, so that it can continue to distinguish gradations in the neuron's response. Thus, $q(t)$ may prove useful for constructing models of the input-output relationship of a spiking neuron (Berry, 1998).

**Acknowledgments**

We would like to thank Mike DeWeese for many useful conversations. One of us, MJB, acknowledges the support of the National Eye Institute. The other, MM, acknowledges the support of the National Science Foundation.

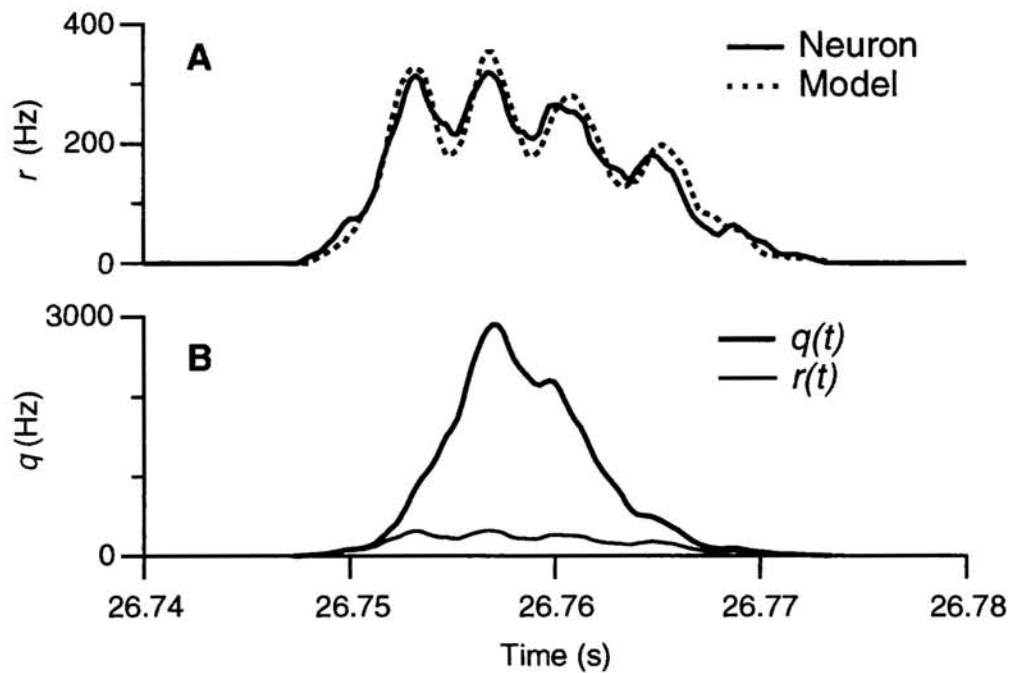

Figure 4: Illustration of the free firing rate. (A) The observed firing rate $r(t)$ for real data (solid) is compared to that from the model (dotted). (B) The free rate $q(t)$ (thick) is shown on the same scale as $r(t)$ (thin). All rates used a time bin of 0.25 ms and boxcar smoothing over 9 bins.

## References

Berry, M. J., D. K. Warland, and M. Meister, *The Structure and Precision of Retinal Spike Trains.* PNAS, USA, 1997. **94**: pp. 5411-5416.

Berry II, M. J. and Markus Meister, *Refractoriness and Neural Precision.* J. Neurosci., 1998. in press.

De Ruyter van Steveninck, R. R., G. D. Lewen, S. P. Strong, R. Koberle, and W. Bialek, *Reliability and Variability in Neural Spike Trains.* Science, 1997. **275**: pp. 1805-1808.

Johnson, D. H. and A. Swami, *The Transmission of Signals by Auditory-Nerve Fiber Discharge Patterns.* J. Acoust. Soc. Am., 1983. **74**: pp. 493-501.

Meister, M., J. Pine, and D. A. Baylor, *Multi-Neuronal Signals from the Retina: Acquisition and Analysis.* J. Neurosci. Methods, 1994. **51**: pp. 95-106.

Miller, M. I. *Algorithms for Removing Recovery-Related Distortion gtom Auditory-Nerve Discharge Patterns.* J. Acoust. Soc. Am., 1985. **77**: pp. 1452-1464.

Rieke, F., D. K. Warland, R. R. de Ruyter van Steveninck, and W. Bialek, *Spikes: Exploring the Neural Code.* 1997, Cambridge, MA: MIT Press.
